# Features as Sufficient Statistics

**D. Geiger** [*]
Department of Computer Science
Courant Institute
and Center for Neural Science
New York University
geiger@cs.nyu.edu

**A. Rudra** [†]
Department of Computer Science
Courant Institute
New York University
archi@cs.nyu.edu

**L. Maloney** [‡]
Departments of Psychology and Neural Science
New York University
ltm@cns.nyu.edu

## Abstract

An image is often represented by a set of detected features. We get an enormous compression by representing images in this way. Furthermore, we get a representation which is little affected by small amounts of noise in the image. However, features are typically chosen in an ad hoc manner. We show how a good set of features can be obtained using sufficient statistics. The idea of sparse data representation naturally arises. We treat the 1-dimensional and 2-dimensional signal reconstruction problem to make our ideas concrete.

## 1 Introduction

Consider an image, $I$, that is the result of a stochastic image-formation process. The process depends on the precise state, $f$, of an environment. The image, accordingly, contains information about the environmental state $f$, possibly corrupted by noise. We wish to choose feature vectors $\phi(I)$ derived from the image that summarize this information concerning the environment. We are not otherwise interested in the contents of the image and wish to discard any information concerning the image that does not depend on the environmental state $f$.

[*]Supported by NSF grant 5274883 and AFOSR grants F 49620-96-1-0159 and F 49620-96-1-0028

[†]Partially supported by AFOSR grants F 49620-96-1-0159 and F 49620-96-1-0028

[‡]Supported by NIH grant EY08266

We develop criteria for choosing sets of features (based on information theory and statistical estimation theory) that extract from the image precisely the information concerning the environmental state.

## 2   Image Formation, Sufficient Statistics and Features

As above, the image $I$ is the realization of a random process with distribution $P_{Environment}(f)$. We are interested in estimating the parameters $f$ of the environmental model given the image (compare [4]). We assume in the sequel that $f$, the environmental parameters, are themselves a random vector with known prior distribution. Let $\phi(I)$ denote a feature vector derived from the the image $I$. Initially, we assume that $\phi(I)$ is a deterministic function of $I$.

For any choice of random variables, $X$, $Y$, define[2] the *mutual information* of $X$ and $Y$ to be $M(X;Y) = \sum_{X,Y} P(X,Y) log \frac{P(X,Y)}{P(X)P(Y)}$. The information about the environmental parameters contained in the image is then $M(f;I)$, while the information about the environmental parameters contained in the feature vector $\phi(I)$ is then $M(f;\phi(I))$. As a consequence of the *data processing inequality*[2], $M(f;\phi(I)) \leq M(f;I)$.

A vector $\phi(I)$, of features is defined to be *sufficient* if the inequality above is an equality. We will use the terms *feature* and *statistic* interchangeably. The definition of a sufficient feature vector above is then just the usual definition of a set of *jointly sufficient statistics*[2].

To summarize, a feature vector $\phi(I)$ captures all the information about the environmental state parameters $f$ precisely when it is suffcient. [1]

**Graded Sufficiency:** A feature vector either is or is not sufficient. For every possible feature vector $\phi(I)$, we define a measure of its failure to be suffcient: $\mathrm{Suff}(\phi(I)) = M(f;I) - M(f;\phi(I))$. This *sufficency measure* is always non-negative and it is zero precisely when $\phi$ is sufficient. We wish to find feature vectors $\phi(I)$ where $\mathrm{Suff}(\phi(I))$ is close to 0. We define $\phi(I)$ to be $\epsilon$-sufficient if $\mathrm{Suff}(\phi(I)) \leq \epsilon$. In what follows, we will ordinarily say sufficient, when we mean $\epsilon$-sufficient.

The above formulation of feature vectors as jointly sufficient statistics, maximizing the mutual information, $M(f, \phi(I))$, can be expressed as the *Kullback-Leibler distance* between the conditional distributions, $P(f|I)$ and $P(f|\phi(I))$:

$$E_I[D(P(f|I) \parallel P(f|\phi(I)))] = M(f;I) - M(f;\phi(I)), \qquad (1)$$

where the symbol $E_I$ denotes the expectation with respect to $I$, $D$ denotes the *Kullback-Leibler (K-L) distance*, defined by $D(f\|g) = \sum_x f(x) \log(f(x)/g(x))$ [2].

Thus, we seek feature vectors $\phi(I)$ such that the conditional distributions, $P(f|I)$ and $P(f|\phi(I))$ in the *K-L* sense, averaged across the set of images. However, this optimization for each image could lead to over-fitting.

## 3   Sparse Data and Sufficient Statistics

The notion of sufficient statistics may be described by how much data can be removed without increasing the *K-L* distance between $P(f|\phi(I))$ and $P(f|I)$. Let us

formulate the approach more precisely, and apply two methods to solve it.

## 3.1   Gaussian Noise Model and Sparse Data

We are required to construct $P(f|I)$ and $P(f|\phi(I))$. Note that according to Bayes'
rule $P(f|\phi(I)) = P(\phi(I)|f) P(f) / P(\phi(I))$. We will assume that the form of
*the model* $P(f)$ is known. In order to obtain $P(\phi(I)|f)$ we write $P(\phi(I)|f) = \sum_I P(\phi(I)|I)P(I|f)$.

**Computing $P(f|\phi(I))$:** Let us first assume that the generative process of the image
$I$, given the model $f$, is Gaussian i.i.d. , i.e., $P(I|f) = (1/\sqrt{2\pi\sigma_i^2}) \prod_i e^{-(f_i-I_i)^2/2\sigma_i^2}$
where $i = 0, 1, ..., N-1$ are the image pixel index for an image of size $N$. Fur-
ther, $P(I_i|f_i)$ is a function of $(I_i - f_i)$ and $I_i$ varies from $-\infty$ to $+\infty$, so that the
normalization constant does not depend on $f_i$. Then, $P(f|I)$ can be obtained by
normalizing $P(f)P(I|f)$.

$$P(f|I) = (1/Z)(\prod_i e^{-(f_i-I_i)^2/2\sigma_i^2})P(f),$$

where $Z$ is the normalization constant.

Let us introduce a binary decision variable $s_i = 0, 1$, which at every image pixel $i$
decides if that image pixel contains "important" information or not regarding the
model $f$. Our statistic $\phi$ is actually a (multivariate) random variable generated
from $I$ according to

$$P_s(\phi|I) = \prod_i \sqrt{\frac{(1-s_i)}{2\pi\sigma_i^2 s_i}} e^{-\frac{1}{2\sigma_i^2}(I_i-\phi_i)^2\frac{(1-s_i)}{s_i}}.$$

This distribution gives $\phi_i = I_i$ with probability 1 (Dirac delta function) when $s_i = 0$
(data is kept) and gives $\phi_i$ uniformly distributed otherwise ($s_i = 1$, data is removed).
We then have

$$
\begin{aligned}
P_s(\phi|f) &= \int P(\phi, I|f)\,dI = \int P(I|f)\,P_s(\phi|I)\,dI \\
&= \prod_i \frac{1}{\sqrt{2\pi\sigma_i^2}} \int e^{-\frac{1}{2\sigma_i^2}(f_i-I_i)^2} \sqrt{\frac{(1-s_i)}{2\pi\sigma_i^2 s_i}} e^{-\frac{1}{2\sigma_i^2}(I_i-\phi_i)^2\frac{(1-s_i)}{s_i}}\,dI_i \\
&= \prod_i \sqrt{\frac{(1-s_i)}{2\pi\sigma_i^2}} e^{-\frac{1}{2\sigma_i^2}(f_i-\phi_i)^2(1-s_i)}.
\end{aligned}
$$

The conditional distribution of $\phi$ on $f$ satisfies the properties that we mentioned in
connection with the posterior distribution of $f$ on $I$. Thus,

$$P_s(f|\phi) = (1/Z_s)\,P(f)\left(\prod_i e^{-\frac{1}{2\sigma_i^2}(f_i-I_i)^2(1-s_i)}\right) \tag{2}$$

where $Z_s$ is a normalization constant.

It is also plausible to extend this model to non-Gaussian ones, by simply modifying
the quadratic term $(f_i - I_i)^2$ and keeping the sparse data coefficient $(1 - s_i)$.

## 3.2   Two Methods

We can now formulate the problem of finding a feature-set, or finding a sufficient
statistics, in terms of the variables $s_i$ that can remove data. More precisely, we can
find $s$ by minimizing

$$E(s, I) = D(P(f|I) \| P_s(f|\phi(I))) + \lambda \sum_i (1 - s_i). \tag{3}$$

It is clear that the $K$-$L$ distance is minimized when $s_i = 0$ everywhere and all the data is kept. The second term is added on to drive the solution towards a minimal sufficient statistic, where the parameter $\lambda$ has to be estimated. Note that, for $\lambda$ very large, all the data is removed ($s_i = 1$), while for $\lambda = 0$ all the data is kept.

We can further write (3) as

$$
\begin{aligned}
E(s, I) &= \sum_f P(f|I) \log(P(f|I)/P_s(f|\phi(I))) + \lambda \sum_i (1 - s_i) \\
&= \sum_f P(f|I) log\left((Z_s/Z) \prod_i e^{-\frac{1}{2\sigma_i^2}(f_i - I_i)^2(1 - (1 - s_i))}\right) + \lambda \sum_i (1 - s_i) \\
&= log\frac{Z_s}{Z} - E_P[\sum_i \frac{s_i}{2\sigma_i^2}(f_i - I_i)^2)] + \lambda \sum_i (1 - s_i).
\end{aligned}
$$

where $E_P[.]$ denotes the expectation taken with respect to the distribution $P$.

If we let $s_i$ be a continuous variable the minimum $E(s, I)$ will occur when

$$0 = \frac{\partial E}{\partial s_i} = (E_{P_s}[(f_i - I_i)^2] - E_P[(f_i - I_i)^2]) - \lambda. \tag{4}$$

We note that the Hessian matrix

$$H_s[i, j] = \frac{\partial^2 E}{\partial s_i \partial s_j} = E_{P_s}[(f_i - I_i)^2(f_j - I_j)^2] - E_{P_s}[(f_i - I_i)^2] E_{P_s}[(f_j - I_j)^2], \tag{5}$$

is a correlation matrix, i.e., it is positive semi-definite. Consequently, $E(s)$ is convex.

**Continuation Method on $\lambda$:**

In order to solve for the optimal vector $s$ we consider the continuation method on the parameter $\lambda$. We know that $s = 0$, for $\lambda = 0$. Then, taking derivatives of (4) with respect to $\lambda$, we obtain

$$\sum_j \frac{\partial^2 E}{\partial s_i \partial s_j} \frac{\partial s_j}{\partial \lambda} - \frac{\partial^2 E}{\partial \lambda \partial s_i} = 0 \quad \rightarrow \quad \frac{\partial s_j}{\partial \lambda} = \sum_i H_s^{-1}[i, j].$$

It was necessary the Hessian to be invertible, i.e., the continuation method works because $E$ is convex. The computations are expected to be mostly spent on estimating the Hessian matrix, i.e., on computing the averages $E_{P_s}[(f_i - I_i)^2(f_j - I_j)^2]$, $E_{P_s}[(f_i - I_i)^2]$, and $E_{P_s}[(f_j - I_j)^2]$. Sometimes these averages can be exactly computed, for example for one dimensional graph lattices. Otherwise these averages could be estimated via Gibbs sampling.

The above method can be very slow, since these computations for $H_s$ have to be repeated at each increment in $\lambda$. We then investigate an alternative direct method.

**A Direct Method:**

Our approach seeks to find a "large set" of $s_i = 1$ and to maintain a distribution $P_s(f|\phi(I))$ close to $P(f|I)$, i.e., to remove as many data points as possible. For this

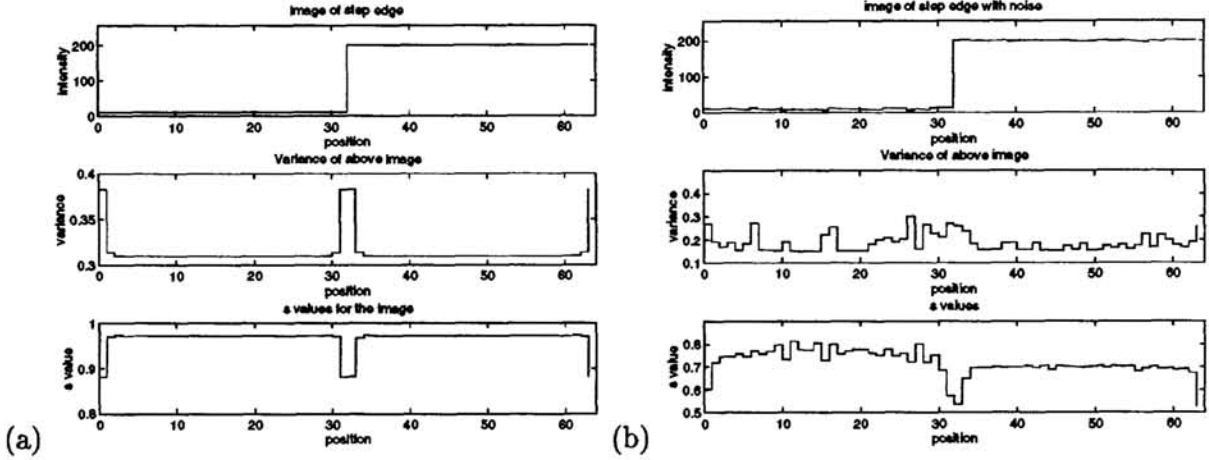

Figure 1: (a). Complete results for step edge showing the image, the effective variance and the computed s-value (using the continuation method). (b) Complete results for step edge with added noise.

goal, we can investigate the marginal distribution

$$
\begin{aligned}
P(f_i|I) &= \int df_0 \dots df_{i-1}\, df_{i+1} \dots df_{N-1}\, P(f|I) \\
&= \frac{1}{Z}\, e^{-\frac{1}{2\sigma_i^2}(f_i-I_i)^2} \int \prod_{j\neq i} df_j\, P(f) \left(\prod_{j\neq i} e^{-\frac{1}{2\sigma_j^2}(f_j-I_j)^2}\right) \\
&= P_{I_i}(f_i)\, P_{eff}(f_i), \quad \text{(after rearranging the normalization constants)}
\end{aligned}
$$

where $P_{eff}(f_i)$ is an effective marginal distribution that depends on all the other values of $I$ besides the one at pixel $i$.

How to decide if $s_i = 0$ or $s_i = 1$ directly from this marginal distribution $P(f_i|I)$? The entropy of the first term $H_{I_i}(f_i) = \int df_i P_{I_i}(f_i)\, log P_{I_i}(f_i)$ indicates how much $f_i$ is conditioned by the data. The larger the entropy the less the data constrain $f_i$, thus, there is less need to keep this data. The second term entropy $H_{eff}(f_i) = \int df_i P_{eff}(f_i)\, log P_{eff}(f_i)$ works the opposite direction. The more $f_i$ is constrained by the neighbors, the lesser the entropy and the lesser the need to keep that data point. Thus, the decision to keep the data, $s_i = 0$, is driven by minimizing the "data" entropy $H_I(f_i)$ and maximizing the neighbor entropy $H_{eff}(f_i)$. The relevant quantity is $H_{eff}(f_i) - H_{I_i}(f_i)$. When this is large, the pixel is kept. Later, we will see a case where the second term is constant, and so the effective entropy is maximized.

For Gaussian models, the entropy is the logarithm of the variance and the appropriate ratio of variances may be considered.

## 4 Example: Surface Reconstruction

To make this approach concrete we apply to the problem of surface reconstruction. First we consider the 1 dimensional case to conclude that edges are the important features. Then, we apply to the two dimensional case to conclude that junctions followed by edges are the important features.

## 4.1 1D Case: Edge Features

Various simplifications and manipulations can be applied for the case that *the model* $f$ is described by a first order Markov model, i.e., $P(f) = \prod_i P_i(f_i, f_{i-1})$. Then the posterior distribution is

$$P(f|I) = \frac{1}{Z} \prod_i e^{-[\frac{1}{2\sigma^2}(f_i - I_i)^2 + \mu_i(f_i - f_{i-1})^2]} ,$$

where $\mu_i$ are smoothing coefficients that may vary from pixel to pixel according to how much intensity change occurs ar pixel $i$, e.g., $\mu_i = \mu \frac{1}{1+\rho(I_i - I_{i-1})^2}$ with $\mu$ and $\rho$ to be estimated. We have assumed that the standard deviation of the noise is homogeneous, to simplify the calculations and analysis of the direct method. Let us now consider both methods, the continuation one and the direct one to estimate the features.

**Continuation Method:** Here we apply $\frac{\partial s_i}{\partial \lambda} = \sum_i H_s^{-1}[i,j]$ by computing $H_s[i,j]$, given by (5), straight forwardly. We use the Baum-Welch method [2] for Markov chains to exactly compute $E_{P_*}[(f_i - I_i)^2(f_j - I_j)^2]$, $E_{P_*}[(f_i - I_i)^2]$, and $E_{P_*}[(f_j - I_j)^2]$. The final result of this algorithm, applied to a step-edge data (and with noise added) is shown in Figure 1. Not surprisingly, the edge data, both pixels, as well as the data boundaries, were the most important data, i.e., the features.

**Direct Method:** We derive the same result, that edges and boundaries are the most important data via an analysis of this model. We use the result that

$$P(f_i|I) = \int df_0 \dots df_{i-1} df_{i+1} \dots df_{N-1} P(f|I) = \frac{1}{Z^N} e^{-\frac{1}{2\sigma^2}(f_i - I_i)^2} e^{-\lambda_i^N(f_i - \Gamma_i^N)^2} ,$$

where $\lambda_i^N$ is obtained recursively, in $\log_2 N$ steps (for simplicity, we are assuming $N$ to be an exact power of 2), as follows

$$\lambda_i^{2K} = (\lambda_i^K + \frac{\lambda_{i+K}^K \mu_{i+K}^K}{\lambda_i^K + \mu_i^K + \mu_{i+K}^K} + \frac{\lambda_{i-K}^K \mu_i^K}{\lambda_i + \mu_i^K + \mu_{i-K}^K}) \qquad (6)$$

where $K \in \{1, 2, 4, 8, \dots, N\}$, $\mu_i^{2K} = \frac{\mu_i^K \mu_{i+K}^K}{\lambda_i^K + \mu_i^K + \mu_{i+K}^K}$, $\Gamma_i^{2K} = \frac{\lambda_i^K \Gamma_i^K + \mu_i^K \Gamma_{i-K}^K + \mu_{i+K}^K \Gamma_{i+K}^K}{\lambda_i^K + \mu_i^K + \mu_{i+K}^K}$, and $\lambda_i^1 = 1/(2\sigma^2)$, $\mu_i^1 = \mu_i$, $\Gamma_i^1 = I_i \quad \forall i$.

The effective variance is given by $var_{eff}(f_i) = 1/(2\lambda_i^N)$ while the data variance is given by $var_I(f_i) = \sigma^2$. Since $var_I(f_i)$ does not depend on any pixel $i$, maximizing the ratio $var_{eff}/var_I$ (as the direct method suggested) as equivalent to maximizing either the effective variance, or the total variance (see figure(1).

Thus, the lower is $\lambda_i^N$ the lower is $s_i$. We note that $\lambda_i^K$ increases with $K$, and $\mu_i^K$ decreases with $K$. Consequently $\lambda^K$ increases less and less as $K$ increases. In a perturbative sense $\lambda_i^2$ most contribute to $\lambda_i^N$ and is defined by the two neighbors values $\mu_i$ and $\mu_{i+1}$, i.e., by the edge information. The larger are the intensity edges the smaller are $\mu_i$ and therefore, the smaller will $\lambda_i^2$ be. Moreover, $\lambda_i^N$ is mostly defined by $\lambda_i^2$ (in a perturbative sense, this is where most of the contribution comes). Thus, we can argue that the pixels $i$ with intensity edges will have smaller values for $\lambda_i^N$ and therefore are likely to have the data kept as a feature ($s_i = 0$).

## 4.2 2D Case: Junctions, Corners, and Edge Features

Let us investigate the two dimensional version of the 1D problem for surface reconstruction. Let us assume the posterior

$$P(f|I) = \frac{1}{Z} e^{-[\frac{1}{2\sigma^2}(f_{ij} - I_{ij})^2 + \mu_{ij}^v(f_{ij} - f_{i-1,j})^2 + \mu_{ij}^h(f_{ij} - f_{i,j-1})^2]} ,$$

where $\mu_{ij}^{v,h}$ are the smoothing coefficients along vertical and horizontal direction, that vary inversely according to the $\nabla I$ along these direction. We can then approximately compute (e.g., see [3])

$$P(f_{ij}|I) \approx \frac{1}{Z} e^{-\frac{1}{2\sigma^2}(f_{ij}-I_{ij})^2} e^{-\lambda_{ij}^N(f_{ij}-\Gamma_{ij}^N)^2}$$

where, analogously to the 1D case, we have

$$\lambda_{ij}^{2K} = \lambda_{ij}^K + \frac{\lambda_{i,j-K}^K \mu_{ij}^{h,K}}{\chi_{i,j-K}^K} + \frac{\lambda_{i,j+K}^K \mu_{i,j+K}^{h,K}}{\chi_{i,j+K}^K} + \frac{\lambda_{i-K,j}^K \mu_{ij}^{v,K}}{\chi_{i-K,j}^K} + \frac{\lambda_{i+K,j}^K \mu_{i+K,j}^{v,K}}{\chi_{i+K,j}^K} \quad (7)$$

where $\chi_{i,j}^K = \lambda_{ij}^K + \mu_{ij}^{h,K} + \mu_{ij}^{v,K} + \mu_{i,j+K}^{h,K} + \mu_{i+K,j}^{v,K}$, and $\mu_{ij}^{h,2K} = \frac{\mu_{ij}^{h,K} \mu_{i,j+K}^{h,K}}{\chi_{i,j}^K}$

The larger is the effective variance at one site $(i,j)$, the smaller is $\lambda^N$, the more likely that image portion to be a feature. The larger the intensity gradient along $h, v$, at $(i,j)$, the smaller $\mu_{ij}^{h,v}$. The smaller is $\mu_{ij}^{h,v}$ the smaller will be contribution to $\lambda^2$. In a perturbative sense ([3]) $\lambda^2$ makes the largest contribution to $\lambda^N$. Thus, at one site, the more intensity edges it has the larger will be the effective variance. Thus, T-junctions will produce very large effective variances, followed by corners, followed by edges. These will be, in order of importance, the features selected to reconstruct 2D surfaces.

## 5  Conclusion

We have proposed an approach to specify when a feature set has sufficient information in them, so that we can represent the image using it. Thus, one can, in principle, tell what kind of feature is likely to be important in a given model. Two methods of computation have been proposed and a concrete analysis for a simple surface reconstruction was carried out.

## Footnotes

[1]An information-theoretic framework has been adopted in neural networks by others; e.g., [5] [9][6] [1][8]. However, the connection between features and sufficiency is new.

[2]We won't prove the result here. The proof is simple and uses the Markov chain property to say that $P(f, I, \phi(I)) = P(I, \phi(I))P(f|I, \phi(I)) = P(I)P(f|I)$.

## References

[1] A. Berger and S. Della Pietra and V. Della Pietra "A Maximum Entropy Approach to Natural Language Processing" *Computational Linguistics*, Vol.22 (1), pp 39–71, 1996.

[2] T. Cover and J. Thomas. *Elements of Information Theory*. Wiley Interscience, New York, 1991.

[3] D. Geiger and J. E. Kogler. Scaling Images and Image Feature via the Renormalization Group. In *Proc. IEEE Conf. on Computer Vision & Pattern Recognition* , New York, NY, 1993.

[4] G. Hinton and Z. Ghahramani. Generative Models for Discovering Sparse Distributed Representations To Appear *Phil. Trans. of the Royal Society* B, 1997.

[5] R. Linsker. Self-Organization in a Perceptual Network. *Computer*, March 1988, 105-117.

[6] J. Principe, U. of Florida at Gainesville Personal Communication

[7] T. Sejnowski. Computational Models and the Development of Topographic Projections *Trends Neurosci*, 10, 304-305.

[8] S.C. Zhu, Y.N. Wu, D. Mumford. Minimax entropy principle and its application to texture modeling *Neural Computation* 1996 B.

[9] P. Viola and W.M. Wells III. "Alignment by Maximization of Mutual Information". In *Proceedings of the International Conference on Computer Vision*. Boston. 1995.